# Better Generative Models for Sequential Data Problems: Bidirectional Recurrent Mixture Density Networks

**Mike Schuster**
ATR Interpreting Telecommunications Research Laboratories
2-2 Hikaridai, Seika-cho, Soraku-gun, Kyoto 619-02, JAPAN
*gustl@itl.atr.co.jp*

## Abstract

This paper describes bidirectional recurrent mixture density networks, which can model multi-modal distributions of the type $P(\mathbf{x}_t|\mathbf{y}_1^T)$ and $P(\mathbf{x}_t|\mathbf{x}_1, \mathbf{x}_2, \ldots, \mathbf{x}_{t-1}, \mathbf{y}_1^T)$ without any explicit assumptions about the use of context. These expressions occur frequently in pattern recognition problems with sequential data, for example in speech recognition. Experiments show that the proposed generative models give a higher likelihood on test data compared to a traditional modeling approach, indicating that they can summarize the statistical properties of the data better.

## 1 Introduction

Many problems of engineering interest can be formulated as sequential data problems in an abstract sense as *supervised learning from sequential data*, where an input vector (dimensionality $D$) sequence $\mathbf{X} = \mathbf{x}_1^T = \{\mathbf{x}_1, \mathbf{x}_2, \ldots, \mathbf{x}_{T-1}, \mathbf{x}_T\}$ living in space $\mathcal{X}$ has to be mapped to an output vector (dimensionality $K$) target sequence $\mathbf{T} = \mathbf{t}_1^T = \{\mathbf{t}_1, \mathbf{t}_2, \ldots, \mathbf{t}_{T-1}, \mathbf{t}_T\}$ in space[1] $\mathcal{Y}$, that often embodies correlations between neighboring vectors $\mathbf{x}_t, \mathbf{x}_{t+1}$ and $\mathbf{t}_t, \mathbf{t}_{t+1}$. In general there are a number of training data sequence pairs (input and target), which are used to estimate the parameters of a given model structure, whose performance can then be evaluated on another set of test data pairs. For many applications the problem becomes to *predict* the *best* sequence $\mathbf{Y}^\star$ given an arbitrary input sequence $\mathbf{X}$, with *'best'* meaning the sequence that minimizes an error using a suitable metric that is yet to be defined. Making use of the theory of pattern recognition [2] this problem is often simplified by treating any sequence as one pattern. This makes it possible to express the objective of sequence prediction with the well known expression $\mathbf{Y}^\star = \arg\max_{\mathcal{Y}} P(\mathbf{Y}|\mathbf{X})$, with $\mathbf{X}$ being the input sequence, $\mathbf{Y}$ being any valid output sequence and $\mathbf{Y}^\star$ being the predicted sequence with the highest probability[2]

among all possible sequences.

*Training* of a sequence prediction system corresponds to estimating the distribution [3] $P(\mathbf{Y}|\mathbf{X})$ from a number of samples which includes (a) defining an appropriate model representing this distribution and (b) estimating its parameters such that $P(\mathbf{Y}|\mathbf{X})$ for the training data is maximized. In practice the model consists of several modules with each of them being responsible for a different part of $P(\mathbf{Y}|\mathbf{X})$.

*Testing* (usage) of the trained system or *recognition* for a given input sequence $\mathbf{X}$ corresponds principally to the evaluation of $P(\mathbf{Y}|\mathbf{X})$ for all possible output sequences to find the best one $\mathbf{Y}^\star$. This procedure is called the *search* and its efficient implementation is important for many applications.

In order to build a model to predict sequences it is necessary to decompose the sequences such that modules responsible for smaller parts can be build. An often used approach is the decomposition into a generative and prior model part, using $P(B|A) = P(A|B)P(B)/P(A)$ and $P(A,B) = P(A)P(B|A)$, as:

$$\mathbf{Y}^\star = \arg\max_{y} P(\mathbf{Y}|\mathbf{X}) = \arg\max_{y} P(\mathbf{X}|\mathbf{Y})P(\mathbf{Y})$$

$$= \arg\max_{y} \underbrace{\left[\prod_{t=1}^{T} P(\mathbf{x}_t|\mathbf{x}_1,\mathbf{x}_2,\ldots,\mathbf{x}_{t-1},\mathbf{y}_1^T)\right]}_{\text{generative part}} \underbrace{\left[\prod_{t=1}^{T} P(\mathbf{y}_t|\mathbf{y}_1,\mathbf{y}_2,\ldots,\mathbf{y}_{t-1})\right]}_{\text{prior part}} \quad (1)$$

For many applications (1) is approximated by simpler expressions, for example as a first order Markov Model

$$\mathbf{Y}^\star \approx \arg\max_{y} \left[\prod_{t=1}^{T} P(\mathbf{x}_t|\mathbf{y}_t)\right]\left[\prod_{t=1}^{T} P(\mathbf{y}_t|\mathbf{y}_{t-1})\right] \quad (2)$$

making some simplifying approximations. These are for this example:

- Every output $\mathbf{y}_t$ depends only on the previous output $\mathbf{y}_{t-1}$ and not on all previous outputs:
$$P(\mathbf{y}_t|\mathbf{y}_1,\mathbf{y}_2,\ldots,\mathbf{y}_{t-1}) \Rightarrow P(\mathbf{y}_t|\mathbf{y}_{t-1}) \quad (3)$$

- The inputs are assumed to be statistically independent in time:
$$P(\mathbf{x}_t|\mathbf{x}_1,\mathbf{x}_2,\ldots,\mathbf{x}_{t-1},\mathbf{y}_1^T) \Rightarrow P(\mathbf{x}_t|\mathbf{y}_1^T) \quad (4)$$

- The likelihood of an input vector $x_t$ given the complete output sequence $\mathbf{y}_1^T$ is assumed to depend only on the output found at $t$ and not on any other ones:
$$P(\mathbf{x}_t|\mathbf{y}_1^T) \Rightarrow P(\mathbf{x}_t|\mathbf{y}_t) \quad (5)$$

Assuming that the output sequences are categorical sequences (consisting of symbols), approximation (2) and derived expressions are the basis for many applications. For example, using Gaussian mixture distributions to model $P(\mathbf{x}_t|\mathbf{y}_t) = P_k(\mathbf{x}) \; \forall \; K$ occuring symbols, approach (2) is used in a more sophisticated form in most state-of-the-art speech recognition systems.

Focus of this paper is to present some models for the generative part of (1) which need less assumptions. Ideally this means to be able to model directly expressions of the form $P(\mathbf{x}_t|\mathbf{x}_1,\mathbf{x}_2,\ldots,\mathbf{x}_{t-1},\mathbf{y}_1^T)$, the possibly (multi-modal) distribution of a vector conditioned on previous $\mathbf{x}$ vectors $\mathbf{x}_t,\mathbf{x}_{t-1},\ldots,\mathbf{x}_1$ and a complete sequence $\mathbf{y}_1^T$, as shown in the next section.

## 2    Mixture density recurrent neural networks

Assume we want to model a continuous vector sequence, conditioned on a sequence of categorical variables as shown in Figure 1. One approach is to assume that the vector sequence can be modeled by a uni-modal Gaussian distribution with a constant variance, making it a uni-modal regression problem. There are many practical examples where this assumption doesn't hold, requiring a more complex output distribution to model multi-modal data. One example is the attempt to model the sounds of phonemes based on data from multiple speakers. A certain phoneme will sound completely different depending on its phonetic environment or on the speaker, and using a single Gaussian with a constant variance would lead to a crude averaging of all examples.

The traditional approach is to build generative models for each symbol separately, as suggested by (2). If conventional Gaussian mixtures are used to model the observed input vectors, then the parameters of the distribution (means, covariances, mixture weights) in general do not change with the temporal position of the vector to model within a given state segment of that symbol. This can be a bad representation for the data in some areas (shown are here the means of a very bi-modal looking distribution), as indicated by the two shown variances for the state 'E'. When used to model speech, a procedure often used to cope with this problem is to increase the number of symbols by grouping often appearing symbol sub-strings into a new symbol and by subdividing each original symbol into a number of states.

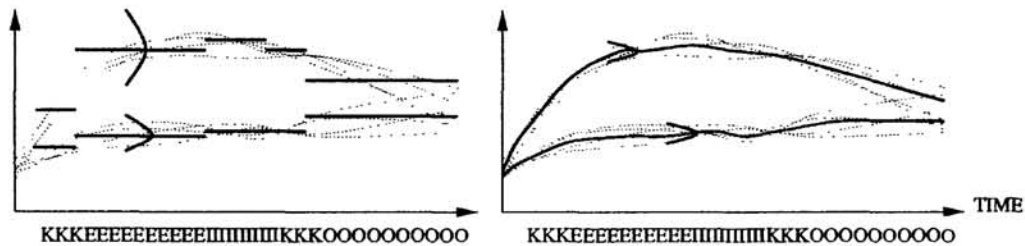

Figure 1: Conventional Gaussian mixtures (left) and mixture density BRNNs (right) for multi-modal regression

Another alternative is explored here, where all parameters of a Gaussian mixture distribution modeling the continuous targets are predicted by one bidirectional recurrent neural network, extended to model mixture densities conditioned on a complete vector sequence, as shown on the right side of Figure 1. Another extension (section 2.1) to the architecture allows the estimation of time varying mixture densities conditioned on a hypothesized output sequence *and* a continuous vector sequence to model exactly the generative term in (1) without *any* explicit approximations about the use of context.

Basics of non-recurrent mixture density networks (MLP type) can be found in [1][2]. The extension from uni-modal to multi-modal regression is somewhat involved but straightforward for the two interesting cases of having a radial covariance matrix or a diagonal covariance matrix per mixture component. They are trained with gradient-descent procedures as regular uni-modal regression NNs. Suitable equations to calculate the error that is back-propagated can be found in [6] for the two cases mentioned, a derivation for the simple case in [1][2].

Conventional recurrent neural networks (RNNs) can model expressions of the form $P(\mathbf{x}_t|\mathbf{y}_1, \mathbf{y}_2, \ldots, \mathbf{y}_t)$, the distribution of a vector given an input vector plus its past input vectors. *Bidirectional* recurrent neural networks (BRNNs) [5][6] are a simple

extension of conventional RNNs. The extension allows one to model expressions of the form $P(\mathbf{x}_t|\mathbf{y}_1^T)$, the distribution of a vector given an input vector plus its past *and following* input vectors.

## 2.1 Mixture density extension for BRNNs

Here two types of extensions of BRNNs to mixture density networks are considered:

I) An extension to model expressions of the type $P(\mathbf{x}_t|\mathbf{y}_1^T)$, a multi-modal distribution of a continuous vector conditioned on a vector sequence $\mathbf{y}_1^T$, here labeled as mixture density BRNN of *Type I*.

II) An extension to model expressions of the type $P(\mathbf{x}_t|\mathbf{x}_1,\mathbf{x}_2,\ldots,\mathbf{x}_{t-1},\mathbf{y}_1^T)$, a probability distribution of a continuous vector conditioned on a vector sequence $\mathbf{y}_1^T$ *and* on its previous context in time $\mathbf{x}_1,\mathbf{x}_2,\ldots,\mathbf{x}_{t-1}$. This architecture is labeled as mixture density BRNN of *Type II*.

The first extension of conventional uni-modal regression BRNNs to mixture density networks is not particularly difficult compared to the non-recurrent implementation, because the changes to model multi-modal distributions are completely independent of the structural changes that have to be made to form a BRNN.

The second extension involves a structural change to the basic BRNN structure to incorporate the $\mathbf{x}_1,\mathbf{x}_2,\ldots,\mathbf{x}_{t-1}$ as additional inputs, as shown in Figure 2. For any $t$ the neighboring $\mathbf{x}_{t-1},\mathbf{x}_{t-2},\ldots$ are incorporated by adding an additional set of weights to feed the hidden forward states with the extended inputs (the targets for the outputs) from the time step before. This includes $\mathbf{x}_{t-1}$ directly and $\mathbf{x}_{t-2},\mathbf{x}_{t-3},\ldots\mathbf{x}_1$ indirectly through the hidden forward neurons. This architecture allows one to estimate the generative term in (1) without making the explicit assumptions (4) and (5), since all the information $\mathbf{x}_t$ is conditioned on, is theoretically available.

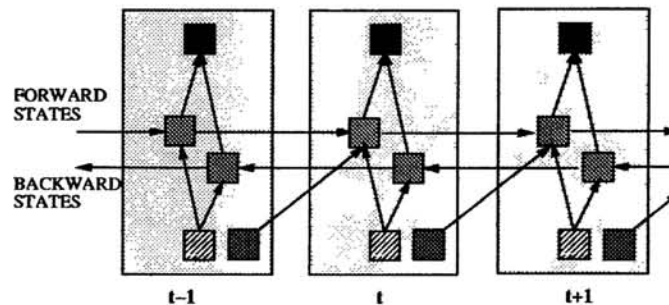

Figure 2: BRNN mixture density extension (Type II) (inputs: striped, outputs: black, hidden neurons: grey, additional inputs: dark grey). Note that without the backward states and the additional inputs this structure is a conventional RNN, unfolded in time.

Different from non-recurrent mixture density networks, the extended BRNNs can predict the parameters of a Gaussian mixture distribution conditioned on a vector *sequence* rather than a single vector, that is, at each (time) position $t$ one parameter set (means, variances (actually standard variations), mixture weights) conditioned on $\mathbf{y}_1^T$ for the BRNN of type I and on $\mathbf{x}_1,\mathbf{x}_2,\ldots,\mathbf{x}_{t-1},\mathbf{y}_1^T$ for the BRNN of type II.

# 3 Experiments and Results

The goal of the experiments is to show that the proposed models are more suitable to model speech data than traditional approaches, because they rely on fewer assumptions. The speech data used here has observation vector sequences representing the original waveform in a compressed form, where each vector is mapped to exactly one out of $K$ phonemes. Here three approaches are compared, which allow the estimation of the likelihood $P(\mathbf{X}|\mathbf{Y})$ with various degrees of approximations:

**Conventional Gaussian mixture model, $P(\mathbf{X}|\mathbf{Y}) \approx \prod_{t=1}^{T} P(\mathbf{x}_t|\mathbf{y}_t)$:**
According to (2) the likelihood of a phoneme class vector is approximated by a conventional Gaussian mixture distribution, that is, a separate mixture model is built to estimate $P(\mathbf{x}|\mathbf{y}) = P_k(\mathbf{x})$ for each of the possible $K$ categorical states in $\mathcal{Y}$. In this case the two assumptions (4) and (5) are necessary. For the variance a radial covariance matrix (diagonal single variance for all vector components) is chosen to match it to the conditions for the BRNN cases below. The number of parameters for the complete model is $KM(D + 2)$ for $M > 1$. Several models of different complexity were trained (Table 1).

**Mixture density BRNN I, $P(\mathbf{X}|\mathbf{Y}) \approx \prod_{t=1}^{T} P(\mathbf{x}_t|\mathbf{y}_1^T)$:** One mixture density BRNN of type I, with the same number of mixture components and a radial covariance matrix for its output distribution as in the approach above, is trained by presenting complete sample sequences to it. Note that for type I all possible context-dependencies (assumption (5)) are automatically taken care of, because the probability is conditioned on complete sequences $\mathbf{y}_1^T$. The sequence $\mathbf{y}_1^T$ contains for any $t$ not only the information about neighboring phonemes, but also the position of a frame within a phoneme. In conventional systems this can only be modeled crudely by introducing a certain number of states per phoneme. The number of outputs for the network depends on the number of mixture components and is $M(D + 2)$. The total number of parameters can be adjusted by changing the number of hidden forward and backward state neurons, and was set here to 64 each.

**Mixture density BRNN II, $P(\mathbf{X}|\mathbf{Y}) = \prod_{t=1}^{T} P(\mathbf{x}_t|\mathbf{x}_1, \mathbf{x}_2, \ldots, \mathbf{x}_{t-1}, \mathbf{y}_1^T)$:**
One mixture density BRNN of type II, again with the same number of mixture components and a radial covariance matrix, is trained under the same conditions as above. Note that in this case both assumptions (4) and (5) are taken care of, because exactly expressions of the required form can be modeled by a mixture density BRNN of type II.

## 3.1 Experiments

The recommended training and test data of the TIMIT speech database [3] was used for the experiments. The TIMIT database comes with hand-aligned phonetic transcriptions for all utterances, which were transformed to sequences of categorical class numbers (training = 702438, test = 256617 vec.). The number of possible categorical classes is the number of phonemes, $K = 61$. The categorical data (input data for the BRNNs) is represented as $K$-dimensional vectors with the $k$th component being one and all others zero. The feature extraction for the waveforms, which resulted in the vector sequences $\mathbf{x}_1^T$ to model, was done as in most speech recognition systems [7]. The variances were normalized with respect to all training data, such that a radial variance for each mixture component in the model is a reasonable choice.

All three model types were trained with $M = 1, 2, 3, 4$, the conventional Gaussian mixture model also with $M = 8, 16$ mixture components. The number of resulting parameters, used as a rough complexity measure for the models, is shown in Table 1. The states of the triphone models were not clustered.

Table 1: Number of parameters for different types of models

| mixture components | mono61 1-state | mono61 3-state | tri571 3-state | BRNN I | BRNN II |
|---|---|---|---|---|---|
| 1 | 1952 | 5856 | 54816 | 20256 | 22176 |
| 2 | 3904 | 11712 | 109632 | 24384 | 26304 |
| 3 | 5856 | 17568 | 164448 | 28512 | 30432 |
| 4 | 7808 | 23424 | 219264 | 32640 | 34560 |
| 8 | 15616 | 46848 | 438528 | – | – |
| 16 | 31232 | 93696 | 877056 | – | – |

Training for the conventional approach using $M$ mixtures of Gaussians was done using the EM algorithm. For some classes with only a few samples $M$ had to be reduced to reach a stationary point of the likelihood. Training of the BRNNs of both types must be done using a gradient descent algorithm. Here a modified version of RPROP [4] was used, which is in more detail described in [6].

The measure used in comparing the tested approaches is the log-likelihood of training and test data given the models built on the training data. In absence of a search algorithm to perform recognition this is a valid measure to evaluate the models since maximizing log-likelihood on the training data is the objective for all model types. Note that the given alignment of vectors to phoneme classes for the test data is used in calculating the log-likelihood on the test data – there is no search for the best alignment.

## 3.2   Results

Figure 3 shows the average log-likelihoods depending on the number of mixture components for all tested approaches on training (upper line) and test data (lower line). The baseline 1-state monophones give the lowest likelihood. The 3-state monophones are slightly better, but have a larger gap between training and test data likelihood. For comparison on the training data a system with 571 distinct triphones with 3 states each was trained also. Note that this system has a lot more parameters than the BRNN systems (see Table 1) it was compared to. The results for the traditional Gaussian mixture systems show how the models become better by building more detailed models for different (phonetic) context, i.e., by using more states and more context classes.

The mixture density BRNN of type I gives a higher likelihood than the traditional Gaussian mixture models. This was expected because the BRNN type I models are, in contrast to the traditional Gaussian mixture models, able to include all possible phonetic context effects by removing assumption (5) – i.e. a frame of a certain phoneme surrounded by frames of any other phonemes with theoretically no restriction about the range of the contextual influence.

The mixture density BRNN of type II, which in addition removes the independence assumption (4), gives a significant higher likelihood than all other models. Note that the difference in likelihood on training and test data for this model is very small, indicating a useful model for the underlying distribution of the data.

## Footnotes

[1] a sample sequence of the *training target data* is denoted as $\mathbf{T}$, while an output sequence in general is denoted as $\mathbf{Y}$, both live in the output space $\mathcal{Y}$

[2] to simplify notation, random variables and their values, are *not* denoted as different symbols. This means, $P(\mathbf{x}) = P(X = \mathbf{x})$.

[3]there is no distinction made between probability mass and density, usually denoted as $P$ and $p$, respectively. If the quantity to model is categorical, a probability mass is assumed, if it is continuous, a probability density is assumed.
